# Nearest Neighbor based Greedy Coordinate Descent

**Inderjit S. Dhillon**
Department of Computer Science
University of Texas at Austin
inderjit@cs.utexas.edu

**Pradeep Ravikumar**
Department of Computer Science
University of Texas at Austin
pradeepr@cs.utexas.edu

**Ambuj Tewari**
Department of Computer Science
University of Texas at Austin
ambuj@cs.utexas.edu

## Abstract

Increasingly, optimization problems in machine learning, especially those arising from high-dimensional statistical estimation, have a large number of variables. Modern statistical estimators developed over the past decade have statistical or *sample complexity* that depends only weakly on the number of parameters when there is some structure to the problem, such as sparsity. A central question is whether similar advances can be made in their computational complexity as well. In this paper, we propose strategies that indicate that such advances can indeed be made. In particular, we investigate the greedy coordinate descent algorithm, and note that performing the greedy step efficiently weakens the costly dependence on the problem size provided the solution is sparse. We then propose a suite of methods that perform these greedy steps efficiently by a reduction to nearest neighbor search. We also devise a more amenable form of greedy descent for composite non-smooth objectives; as well as several approximate variants of such greedy descent. We develop a practical implementation of our algorithm that combines greedy coordinate descent with locality sensitive hashing. Without tuning the latter data structure, we are not only able to significantly speed up the vanilla greedy method, but also outperform cyclic descent when the problem size becomes large. Our results indicate the effectiveness of our nearest neighbor strategies, and also point to many open questions regarding the development of computational geometric techniques tailored towards first-order optimization methods.

## 1  Introduction

Increasingly, optimization problems in machine learning are very high-dimensional, where the number of variables is very large. This has led to a renewed interest in iterative algorithms that require bounded time per iteration. Such iterative methods take various forms such as so-called row-action methods [6] which enforce constraints in the optimization problem sequentially, or first-order methods [4] which only compute the gradient or a coordinate of the gradient per step. But the overall time complexity of these methods still has a high polynomial dependence on the number of parameters. Modern statistical estimators developed over the past decade have statistical or sample complexity that depends only weakly on the number of parameters [5, 15, 18]. *Can similar advances be made in their computational complexity?*

Towards this, we investigate one of the simplest classes of first order methods: coordinate descent, which only updates a single coordinate of the iterate at every step. The coordinate descent class of algorithms has seen a renewed interest after recent papers [8, 10, 19] have shown considerable empirical success in application to large problems. Saha and Tewari [13] even show that under

certain conditions, the convergence rate of cyclic coordinate descent is at least as fast as that of gradient descent.

In this paper, we focus on high-dimensional optimization problems where the solution is sparse. We were motivated to investigate coordinate descent algorithms by the intuition that they could leverage the sparsity structure of the solution by judiciously choosing the coordinate to be updated. In particular, we show that a greedy selection of the coordinates succeeds in weakening the costly dependence on problem size with the caveat that we could perform the greedy step efficiently. The naive greedy updates would however take time that scales at least linearly in the problem dimension $O(p)$ since it has to compute the coordinate with the maximum gradient. We thus come to the other main question of this paper: *Can the greedy steps in a greedy coordinate scheme be performed efficiently?* Surprisingly, we are able to answer in the affirmative, and we show this by a reduction to nearest neighbor search. This allows us to leverage the significant amount of recent research on *sublinear* methods for nearest neighbor search, provided it suffices to have approximate nearest neighbors. The upshot of our results is a suite of methods that depend weakly on the problem size or number of parameters. We also investigate several notions of approximate greedy coordinate descent for which we are able to derive similar rates. For the composite objective case, where the objective is the sum of a smooth component and a separable non-smooth component, we propose and analyze a "look-ahead" variant of greedy coordinate descent.

The development in this paper thus raises *a new line of research* on connections between computational geometry and first-order optimization methods. For instance, given our results, it would be of interest to develop approximate nearest neighbor methods tuned to greedy coordinate descent. As an instance of such a connection, we show that if the covariates underlying the optimization objective satisfy a mutual incoherence condition, then a very simple nearest neighbor data structure suffices to yield a good approximation. Finally, we provide simulations that not only show that greedy coordinate descent with approximate nearest neighbor search performs overwhelmingly better than vanilla greedy coordinate descent, but also that it starts outperforming cyclic descent when the problem size increases: the larger the number of variables, the greater the relative improvement in performance. The results of this paper naturally lead to several open problems: can effective computational geometric data structures be tailored towards greedy coordinate descent? Can these be adapted to (a) other first-order methods, perhaps based on sampling, and (b) different regularized variants that uncover structured sparsity. We hope this paper fosters further research and cross-fertilization of ideas in computational geometry and optimization.

## 2 Setup and Notation

We start our treatment with differentiable objective functions, and then extend this to encompass non-differentiable functions which arise as the sum of a smooth component and a separable non-smooth component. Let $\mathcal{L} : \mathbb{R}^p \to \mathbb{R}$ be a convex differentiable function. We do not assume that the function is strongly convex: indeed most optimizations arising out of high-dimensional machine learning problems are convex but typically not strongly so. Our analysis requires that the function satisfies the following coordinate-wise Lipschitz condition:

**Assumption A1.** The loss function $\mathcal{L}$ satisfies
$$\|\nabla\mathcal{L}(w) - \nabla\mathcal{L}(v)\|_\infty \leq \kappa_1 \cdot \|w - v\|_1, \text{ for some } \kappa_1 > 0.$$
We note that this condition is weaker than the standard Lipschitz conditions on the gradients. In particular, we say that $\mathcal{L}$ has $\kappa_2$-Lipschitz continuous gradient w.r.t. $\|\cdot\|_2$ when $\|\nabla\mathcal{L}(w) - \nabla\mathcal{L}(v)\|_2 \leq \kappa_2 \cdot \|w - v\|_2$. Note that $\kappa_1 \leq \kappa_2$; indeed $\kappa_1$ could be up to $p$ times smaller than $\kappa_2$. E.g. when $\mathcal{L}(w) = 1/2w^\top Aw$ with a positive semi-definite matrix $A$, we have $\kappa_1 = \max_j A_{j,j}$, the maximum entry on the diagonal, while $\kappa_2 = \max_j \lambda_j(A)$, the maxium eigenvalue of $A$. It is thus possible for $\kappa_2$ to be much larger than $\kappa_1$: for instance $\kappa_2 = p\kappa_1$ when $A$ is the all 1's matrix.

We are interested in the general optimization problem,
$$\min_{w \in \mathbb{R}^p} \mathcal{L}(w). \tag{1}$$

We will focus on the case where the solution is bounded and sparse. We thus assume:

**Assumption A2.** The solution $w^*$ of (1) satisfies: $\|w^*\|_\infty \leq B$ for some constant $B < \infty$ independent of $p$, and that $\|w^*\|_0 = s$, i.e., solution is $s$-sparse.

### 2.1 Coordinate Descent

Coordinate descent solves (1) iteratively by optimizing over a single coordinate while holding others fixed. Typically, the choice of the coordinate to be updated is cyclic. One caveat with this scheme

however is that it could be expensive to compute the one-dimensional optimum for general functions $\mathcal{L}$. Moreover when $\mathcal{L}$ is not smooth, such coordinatewise descent is not guaranteed to converge to the global optimum in general, unless the non-differentiable component is separable [16]. A line of recent work [16, 17, 14] has thus focused on a "gradient descent" version of coordinate descent, that iteratively uses a local quadratic upper bound $\mathcal{L}^U$ of the function $\mathcal{L}$. For the case where the optimization function is the sum of a smooth function and the $\ell_1$ regularizer, this variant is also called Iterative Soft Thresholding [7]. A template for such coordinate gradient descent is the set of iterates: $w^t = w^{t-1} - \frac{1}{\kappa_1} \nabla_j \mathcal{L}(w^t) e_j$. Friedman et al. [8], Genkin et al. [10], Wu and Lange [19] and others have shown considerable empirical success in applying these to large problems.

### 2.2 Greedy Coordinate Descent

In this section, we focus on a simple deterministic variant of coordinate descent that picks the coordinate that attains the coordinatewise maximum of the gradient vector:

---
**Algorithm 1** Greedy Coordinate Gradient Descent

---
Initialize: Set the initial value of $w^0$.
**for** $t = 1, \ldots$ **do**
  $j = \arg\max_l |\nabla_l \mathcal{L}(w^t)|$.
  $w^t = w^{t-1} - \frac{1}{\kappa_1} \nabla_j \mathcal{L}(w^t) e_j$.
**end for**

---

**Lemma 1.** Suppose the convex differentiable function $\mathcal{L}$ satisfies Assumptions A1 and A2. Then the iterates of Algorithm 1 satisfy:

$$\mathcal{L}(w^t) - \mathcal{L}(w^*) \leq \frac{\kappa_1}{2} \frac{\|w^0 - w^*\|_1^2}{t}.$$

Letting $c(p)$ denote the time required to solve each greedy step $\max_l |\nabla_l \mathcal{L}(w^t)|$, the greedy version of coordinate descent achieves the rate $\mathcal{L}(w^t) - \mathcal{L}(w^*) = O(s^2 \ c(p)/T)$ at time $T$. Note that the dependence on the problem size $p$ is restricted to the greedy step: if we could solve this maximization more efficiently, then we have a powerful "active-set" method. While brute force maximization for the greedy step would take $O(p)$ time, if it can be done in $O(1)$ time, then at time $T$, the iterate $w$ satisfies $\mathcal{L}(w) - \mathcal{L}(w^*) = O(s^2/T)$ which would be *independent of the problem size*.

## 3 Nearest Neighbor and Fast Greedy

In this section, we examine whether the greedy step can be performed in *sublinear time*. We focus in particular on optimization problems arising from statistical learning problems where the optimization objective can be written as

$$\mathcal{L}(w) = \sum_{i=1}^n \ell(w^T x^i, y^i), \tag{2}$$

for some loss function $\ell : \mathbb{R} \times \mathbb{R} \mapsto \mathbb{R}$, and a set of observations $\{(x^i, y^i)\}_{i=1}^n$, with $x^i \in \mathbb{R}^p, y^i \in \mathbb{R}$. Note that such an optimization objective arises in most statistical learning problems. For instance, consider linear regression, with response $y = \langle w, x \rangle + \epsilon$, where $\epsilon \sim \mathcal{N}(0, 1)$. Then given observations $\{(x^i, y^i)\}_{i=1}^n$, the maximum likelihood problem has the form of (2), with $\ell(u, v) = (u - v)^2$.

Letting $g(u, v) = \nabla_u \ell(u, v)$ denote the gradient of the sample loss with respect to its first argument, and $r^i(w) = g(w^T x^i, y^i)$, the gradient of the objective (2) may be written as $\nabla_j \mathcal{L}(w) = \sum_{i=1}^n x_j^i \ r^i(w) = \langle x_j, r(w) \rangle$. It then follows that the greedy coordinate descent step in Algorithm 1 reduces to the following simple problem:

$$\max_j |\langle x_j, r(w) \rangle|. \tag{3}$$

We can now see why the greedy step (3) can be performed efficiently: it can be cast as a nearness problem. Indeed, assume that the data is standardized so that $\|x_j\| = 1$ for $j = 1, \ldots, p$. Let $\bar{x} = \{x_1, \ldots, x_p, -x_1, \ldots, -x_p\}$ include the negated data vectors. Then, it can be seen that

$$\arg\max_{j \in [p]} |\langle x_j, r \rangle| \equiv \arg\min_{j \in [2p]} \|\bar{x}_j - r\|_2^2. \tag{4}$$

Thus, the greedy step amounts to a nearest neighbor problem of computing the nearest point to $r$ in the set $\{\bar{x}_j\}_{j=1}^{2p}$. While this would take $O(pn)$ time via brute force, the hope is to leverage the state of

the art in nearest neighbor search [11] to perform this greedy selection in sublinear time. Regarding the time taken to compute the gradient $r(w)$, note that after any coordinate descent update, we can update $r^i$ in $O(1)$ time if we cache the values $\{\langle w, x^i \rangle\}$, so that $r$ can be updated in $O(n)$ time.

The reduction to nearest neighbor search however comes with a caveat: nearest neighbor search variants that run in sublinear time only compute *approximate nearest neighbors*. This in turn amounts to performing the greedy step approximately. In the next few subsections, we investigate the consequences of such approximations.

### 3.1 Multiplicative Greedy

We first consider a variant where the greedy step is performed under a multiplicative approximation, where we choose a coordinate $j_t$ such that, for some $c \in (0, 1]$,

$$|[\nabla\mathcal{L}(w^t)]_{j_t}| \geq c \cdot \|\nabla\mathcal{L}(w^t)\|_\infty . \tag{5}$$

As the following lemma shows, the approximate greedy steps have little qualitative effect (proof in Supplementary Material).

**Lemma 2.** The greedy coordinate descent iterates, with the greedy step computed as in (5), satisfy:

$$\mathcal{L}(w^t) - \mathcal{L}(w^\star) \leq \frac{1}{c} \cdot \frac{\kappa_1 \|w^0 - w^\star\|_1^2}{t} .$$

The price for the approximate greedy updates is thus just a constant factor $1/c \geq 1$ reduction in the convergence rate.

Note that the equivalence of (4) need not hold under multiplicative approximations. That is, approximate nearest neighbor techniques that obtain a *nearest neighbor* upto a multiplicative factor, do not guarantee a multiplicative approximation for the inner product in the greedy step in turn. As the next lemma shows this still achieves the required qualitative rate.

**Lemma 3.** Suppose the greedy step is performed as in (5) with a multiplicative approximation factor of $(1 + \epsilon_m)$ (due to approximate nearest neighbor search for instance). Then, at any iteration $t$, the greedy coordinate descent iterates satisfy either of the following two conditions, for any $\epsilon > 0$:

(a) $\nabla\mathcal{L}(w^t)$ is small (i.e. the iterate is near-stationary): $\|\nabla\mathcal{L}(w^t)\|_\infty \leq \frac{\epsilon + \epsilon_m}{(1 + \epsilon_m)}\|r(w^t)\|_2$, or

(b) $\mathcal{L}(w^t) - \mathcal{L}(w^\star) \leq \frac{1 + \epsilon_m}{\epsilon_m(1/\epsilon) + 1} \cdot \frac{\kappa_1 \|w^0 - w^\star\|_1^2}{t}$ .

### 3.2 Additive Greedy

Another natural variant is the following additive approximate greedy coordinate descent, where we choose the coordinate $j_t$ such that

$$|[\nabla\mathcal{L}(w^t)]_{j_t}| \geq \|\nabla\mathcal{L}(w^t)\|_\infty - \epsilon_{add} , \tag{6}$$

for some $\epsilon_{add}$. As the lemma below shows, the approximate greedy steps have little qualitative effect.

**Lemma 4.** The greedy coordinate descent iterates, with the greedy step computed as in (6), satisfy:

$$\mathcal{L}(w^t) - \mathcal{L}(w^\star) \leq \frac{\kappa_1 \|w^0 - w^\star\|_1^2}{t} + \epsilon_{add} .$$

Note that we need obtain an additive approximation in the greedy step only upto the order of the final precision desired of the optimization problem. In particular, for statistical estimation problems the desired optimization accuracy need not be lower than the *statistical precision*, which is typically of the order of $s\log(p)/\sqrt{n}$. Indeed, given the connections elucidated above to greedy coordinate descent, it is an interesting future problem to develop approximate nearest neighbor methods with additive approximations.

## 4 Tailored Nearest Neighbor Data Structures

In this section, we show that one could develop approximate nearest neighbor methods tailored to the statistical estimation setting.

## 4.1 Quadtree under Mutual Incoherence

We will show that just a vanilla quadtree yields a good approximation when the covariates satisfy a technical statistical condition of mutual coherence. A quadtree is a tree data structure which partitions the space. Each internal node $u$ in the quadtree has a representative point, denoted by $\text{rep}(u)$, and a list of children nodes, denoted by $\text{children}(u)$, which partition the space under $u$. For further details, we refer to Har-Peled [11]. The spread $\Phi(D)$ of the set of points $D$ is defined as $\Phi(D) = \frac{\max_{i \neq j} \|x_i - x_j\|}{\min_{i \neq j} \|x_i - x_j\|}$, and is the ratio between the diameter of $D$ and the closest pair distance of points in $D$. Following Har-Peled [11], we can show that the depth of the quadtree by the standard construction is bounded by $O(\log \Phi(D) + \log n)$ and can be constructed in time $O(p \log(n \Phi(D)))$.

Here, we show that a standard nearest neighbor algorithm using quadtrees Har-Peled [11], Arya and Mount [2], rewritten below to allow for arbitrary approximation factor $(1 + \epsilon_{\text{nn}})$, suffices under appropriate statistical conditions.

---

Input: quadtree $T$, approx. factor $(1 + \epsilon_{\text{nn}})$, query $r$.
Initialize: $i = 0$; $A_0 = \{\text{root}(T)\}$.
**while** $A_i \neq \{\}$ **do**
  **for** each node $v \in A_i$ **do**
    $u_{\text{ann}} = \text{nn}(r, \{u_{\text{ann}}\} \cup \text{rep}(\text{children}(v)))$.
    **for** each node $u \in \text{children}(v)$ **do**
      **if** $\|r - \text{rep}(u)\| - \text{diam}(u) < \|r - u_{\text{ann}}\|/(1 + \epsilon_{\text{nn}})$, then $A_{i+1} = A_{i+1} \cup \{u\}$.
    **end for**
  **end for**
  $i \leftarrow i + 1$
**end while**
Return $u_{\text{ann}}$.

---

**Lemma 5.** Let $(1 + \epsilon_{\text{nn}})$ be the approximation factor for the approximate nearest neighbor search. Let $nn(r)$ be the true nearest neighbor to $r$. Then the output $u_{\text{ann}}$ of Algorithm 4.1 satisfies

$$\|r - u_{\text{ann}}\|_2 \leq (1 + \epsilon_{\text{nn}})\|r - nn(r)\|_2.$$

*Proof.* Let $u$ be the last node in the quadtree containing $nn(r)$ thrown away by the algorithm. Then,

$$\|r - nn(r)\| \geq \|r - \text{rep}(u)\| - \|\text{rep}(u) - nn(r)\| \geq \|r - \text{rep}(u)\| - \text{diam}(u) \geq \frac{\|r - u_{\text{ann}}\|}{1 + \epsilon_{\text{nn}}},$$

whence the statement in the theorem follows. $\square$

The next lemma shows the time taken by the algorithm. Again, we rewrite the analysis of Har-Peled [11], Arya and Mount [2] to allow for arbitrary approximation factors.

**Lemma 6.** The time taken by algorithm 4.1 to compute a $(1 + \epsilon_{\text{nn}})$-nearest neighbor to $r$ from $D = \{x_1, \ldots, x_p\}$ is $O\left(\log(\Phi(D)) + \left(1 + \frac{1}{\epsilon_{\text{nn}}}\right)^n\right)$.

As the next lemma shows, the spread is controlled when the mutual coherence of the covariates is small. In particular, define $\mu(D) = \max_{i \neq j} \langle x_i, x_j \rangle$. We require that the mutual coherence $\mu(D)$ be small and in particular be bounded away from 1. Such a condition is typically imposed as sufficient condition for sparse parameter recovery [5, 15]. Intriguingly, this very condition allows us to provide guarantees for optimization. This thus adds to the burgeoning set of recent papers that are finding that conditions imposed for strong statistical guarantees are useful in turn for obtaining faster rates in the corresponding optimization problems.

Under this condition, the closest pair distance can be bounded as, $\|x_i - x_j\|^2 = 2 - 2\langle x_i, x_j \rangle \geq 2(1 - \mu)$, which in turn allows us to control the spread: $\Phi(D) \leq \frac{2}{\sqrt{2(1-\mu)}} = \sqrt{\frac{2}{1-\mu}}$, which thus yields the corollary:

**Lemma 7.** Suppose the mutual coherence of the covariates $D = \{x_1, \ldots, x_p\}$ is bounded so that $\mu(D) < 1$. Then the time taken by algorithm 4.1 to compute a $(1 + \epsilon_{\text{nn}})$-nearest neighbor to $r$ from is $O\left(\log\left(\frac{1}{1-\mu}\right) + \left(1 + \frac{1}{\epsilon_{\text{nn}}}\right)^n\right)$.

While this data structure is quite useful in most settings, it requires that the mutual coherence of the covariates be bounded, and further the time required is exponential (but weakly so) in the number of samples. However, following [1, 11], we can use random projections to bring the runtime down to $O(p^{\epsilon_{nn}^{-2}})$, and the preprocessing time to $O(n\,p\,\log p\,\epsilon_{nn}^{-2})$.

## 5 Overall Time Complexity

In the previous sections, we saw that the greedy step for generalized linear models is equivalent to nearest neighbor search: given any *query* $r$, we want to find its nearest neighbor among the $p$ points $D = \{x_1, \ldots, x_p\}$ each in $\mathbb{R}^n$. Standard data structures include quadtrees which spatially partition the data, and KD trees which partition the data according to their point mass.

Approximate nearest neighbor search [11] estimates an *approximate* nearest neighbor, upto a multiplicative approximation say $\epsilon_{nn}$: so that if the nearest neighbor to $r$ is $x_j$ and the algorithm outputs $x_k$, then it guarantees that $\|x_k - r\|_2 \le (1 + \epsilon_{nn})\|x_j - r\|$. Any such nearest neighbor algorithm, given a query $r$, incurs time depends on the number of points $p$ (typically sublinearly), their dimension $n$, and the approximation factor $(1 + \epsilon_{nn})$. Let us denote this cost by $C_t(n, p, \epsilon_{nn})$.

From our analysis of multiplicative approximate greedy (see Lemma 3), given a multiplicative approximation factor $(1 + \epsilon_{nn})$ in the approximate nearest neighbor method, the approximate greedy coordinate descent has the convergence rate: $\frac{K}{\epsilon_{nn}} \cdot \frac{\kappa_1 s^2}{t}$ for some constant $K > 0$. Thus, the number of iterations required to obtain a solution with accuracy $\epsilon_{opt}$ is given by, $T_{greedy} = \frac{K\kappa_1 s^2}{\epsilon_{nn}\,\epsilon_{opt}}$. Since each of these greedy steps have cost $C_t(n, p, \epsilon_{nn})$, the overall cost $C_G$ is given as: $C_G = C_t(n, p, \epsilon_{nn}) \cdot \frac{K\kappa_1 s^2}{\epsilon_{nn}\,\epsilon_{opt}}$. Of course these approximate nearest neighbor methods also require some pre-processing time $C_-(p, n, \epsilon_{nn})$, but this can typically be amortized across multiple runs of the optimization problem with the same covariates (for a regularization path for instance). It could also be reused across different models, and for other forms of data analysis. Examples include:

**(a).** Locality Sensitive Hashing [12] uses random shifting windows and random projections to hash the data points such that distant points do not collide with high probability. Let $\rho = 1/(1 + \epsilon_{nn}) < 1$. Then here, $C_-(p, n, \epsilon_{nn}) = O\left(n\,p^{1+\rho}\,\epsilon_{nn}^{-2}\right)$ while $C_t(n, p, \epsilon_{nn}) = O(np^\rho)$. Thus, for sparse solutions $s = o(\sqrt{p})$, the runtime cost scales as $C_G = O\left(n\,p^\rho\,\epsilon_{nn}^{-1}\epsilon_{opt}^{-1}\right)$.

**(b).** Ailon and Chazelle [1] use multiple lookup tables after random projections to obtain a nearest neighbor data structure with costs and $C_-(p, n, \epsilon_{nn}) = O(p^{\epsilon_{nn}^{-2}})$, and $C_t(p, n, \epsilon_{nn}) = O(n \log n + \epsilon_{nn}^{-3} \log^2 p)$. Thus the runtime cost here scales as $C_G = O\left(\frac{n \log n + \epsilon_{nn}^{-3} \log^2 p}{\epsilon_{nn}\,\epsilon_{opt}}\right)$.

**(c).** In Section 4, we showed that when the covariates are *mutually incoherent*, then we can use a simple quadtree, and random Gaussian projections to obtain $C_-(p, n, \epsilon_{nn}) = O(n\,p\,\log p\,\epsilon_{nn}^{-2})$ and $C_t(p, n, \epsilon_{nn}) = O(p^{\epsilon_{nn}^{-2}})$. Thus the runtime cost here scales as $C_G = O\left(p^{\epsilon_{nn}^{-2}}\,\epsilon_{opt}^{-1}\epsilon_{nn}^{-1}\right)$.

## 6 Non-Smooth Objectives

Now we consider the more general composite objective case where the objective is the sum of a differentiable, and a separable non-differentiable function:

$$\min_{w \in \mathbb{R}^p} \mathcal{L}(w) + \mathcal{R}(w), \tag{7}$$

where we assume $\mathcal{L}$ is convex and differentiable and satisfies the Lipshitz condition in Assumption A1, and $\mathcal{R}(w) = \sum_j \mathcal{R}_j(w_j)$ where $\mathcal{R}_j : \mathbb{R} \mapsto \mathbb{R}$ could be non-differentiable. Again, we assume that Assumption 2 holds. The natural counterpart of the greedy algorithms in the previous sections would be to pick the coordinate with the maximum absolute value of the *subgradient*. However, we did not observe good performance for this variant either theoretically or in simulations. Thus, we now study a *lookahead* variant that picks the coordinate with the maximum absolute value of the sum of the gradient of the smooth component and the subgradient of the non-smooth component at the next iterate.

Denote $[\nabla \mathcal{L}(w^t)]_j$ by $G_j^t$, and compute the *next* iterate $w_j^{t+1}$ as $\arg\min_w g_j^t(w - w_j^t) + \frac{\kappa_1}{2}(w - w_j^t)^2 + R_j(w)$. Let $\rho_j = \partial R_j(w_j^{t+1})$ denote the subgradient at this next iterate, and let

$$\eta_j^t = (-1/\kappa_1)(g_j^t + \rho_j) = w_j^{t+1} - w_j^t. \tag{8}$$

**Algorithm 2** A Greedy Coordinate Descent Algorithm for Composite Objectives

1: Initialize: $w^0 \leftarrow \mathbf{0}$
2: **for** $t = 1, 2, 3, \ldots$ **do**
3: $\quad j_t \leftarrow \arg\max_{j \in [p]} \left| \eta_j^t \right|$ (with $\eta_j^t$ as defined in (8))
4: $\quad w^{t+1} \leftarrow w^t + \eta_{j_t}^t \mathbf{e}_{j_t}$,
5: **end for**

Then pick the coordinate as $\arg\max_{j \in [p]} \left| \eta_j^t \right|$. The next lemma states that this variant performs qualitatively similar to its smooth counterpart in Algorithm 1.

**Lemma 8.** The greedy coordinate descent iterates of Algorithm 2 satisfy:

$$\mathcal{L}(w^t) + \mathcal{R}(w^t) - \mathcal{L}(w^*) - \mathcal{R}(w^*) \leq \frac{\kappa_1}{2} \frac{\|w^0 - w^*\|_1^2}{t}.$$

The greedy step for composite objectives in Algorithm 2 at any iteration $t$ entails solving the maximization problem: $\max_j |\eta_j^t|$, where $\eta_j$ is as defined in (8). Let us focus on the case where the regularizer $\mathcal{R}$ is the $\ell_1$ norm, so that $\mathcal{R}(w) = \lambda \sum_{j=1}^p |w_j|$, for some $\lambda > 0$. Using the notation from above, we thus have the following objective: $\min_w \frac{1}{n} \sum_{i=1}^n \ell(w^T x_i, y_i) + \lambda \|w\|_1$. Then $\eta_j$ from (8) can be written in this case as: $\eta_j = \mathcal{S}_{\lambda/\kappa_1}(w_j^t - \langle x_j, r(w^t) \rangle / \kappa_1) - w_j^t$, where $\mathcal{S}_r(u) = \text{sign}(u) \max\{|u| - r, 0\}$ is the soft-thresholding function. So the greedy step reduces to maximizing $\max_j |\mathcal{S}_{\lambda/\kappa_1}(w_j^t - \langle x_j, r(w^t) \rangle / \kappa_1) - w_j^t|$ over $j$. The next lemma shows that by focusing the maximization on the inner products $\langle x_j, r(w) \rangle$ we lose at most a factor of $\lambda/\kappa_1$:

**Lemma 9.** $\left| |\langle x_j, r(w^t) \rangle / \kappa_1| - |\eta_j^t| \right| \leq \lambda/\kappa_1$.

The Lemma in turn implies that if $j' \in \arg\max_{j \in [p]} |\langle x_j, r(w^t) \rangle / \kappa_1|$, then

$$|\eta_{j'}^t| \leq |\langle x_{j'}, r(w^t) \rangle / \kappa_1| + \lambda/\kappa_1 = \max_{j \in [p]} |\langle x_j, r(w^t) \rangle / \kappa_1| + \lambda/\kappa_1 \leq \max_{j \in [p]} |\eta_j^t| + 2\lambda/\kappa_1.$$

Typical setting of $\lambda$ for statistical estimation is at the level of the statistical precision of the problem (and indeed of the order of $O(1/\sqrt{n})$ even for low-dimensional problems). Thus, as in the previous section, we estimate the coordinate $j$ that maximizes the inner product $|\langle x_j, r(w) \rangle|$, which in turn can be approximated using approximate nearest neighbor search. So, even for composite objectives, we can reduce the greedy step to performing a nearest neighbor search. Note however that this can be performed sublinearly only at the cost of recovering an *approximate* nearest neighbor. Note that this in turn entails that we would be performing each greedy step in coordinate descent approximately.

## 7 Experimental Results

We conducted speed trials in MATLAB comparing 3 algorithms: greedy (Algorithm 2), greedy.LSH (coordinate to update chosen by LSH) and cyclic on $\ell_1$-regularized problems: $\sum_{i=1}^n \ell(w^T x_i, y_i) + \lambda \|w\|_1$ where $\ell(y, t)$ was either $(y - t)^2/2$ (squared loss) or $\log(1 + \exp(-ty))$ (logistic loss) and we chose $\lambda = 0.01$. All these algorithms, after selecting a coordinate to update, minimize the function fully along that coordinate. For squared loss, this minimum can be obtained in closed form while for logistic we performed 6 steps of the (1-dimensional) Newton method. The data was generated as follows: a matrix $X \in \mathbb{R}^{n,p}$ was chosen with i.i.d. standard normal entries and the each column was normalized to $\ell_2$-norm 1. Then, we set $Y = X w_{\text{tr}}$ for a $k$-sparse vector $w_{\text{tr}} \in \mathbb{R}^p$ (with nonzero entries placed randomly). The labels $y_i$ were chosen to be either $Y_i$ or $\text{sign}(Y_i)$ depending on whether the squared or logistic loss was being optimized. The rows of $X$ became the instances $x_i$.

Figure 1 shows the objective function value versus CPU time plots for the logistic loss with $p = 10^4, 10^5, 10^6$. As $p$ grows we keep $k = 100$ constant and scale $n$ as $\lfloor 4k \log(p) \rfloor$. In this case, greedy.LSH not only speeds up naive greedy significantly but also beats cyclic coordinate descent. In fact, cyclic appears to be stalled especially for $p = 10^5, 10^6$. The reason for this is that cyclic, in the time allotted, was only able to complete $52\%, 40\%$ and $27\%$ of a full sweep through the $p$ coordinates for $p = 10^4, 10^5$ and $10^6$ respectively. Furthermore, cyclic had generated far less sparse final iterates than greedy.LSH in all 3 cases. Figure 2 shows the same plots but for squared loss. Here, since each coordinate minimization is closed form and thus very quick, greedy.LSH has a harder time competing with it. Greedy.LSH is still way faster than naive greedy and start to beat cyclic at $p = 10^6$. The trend of greedy.LSH catching up with cyclic as $p$ grows is clearly

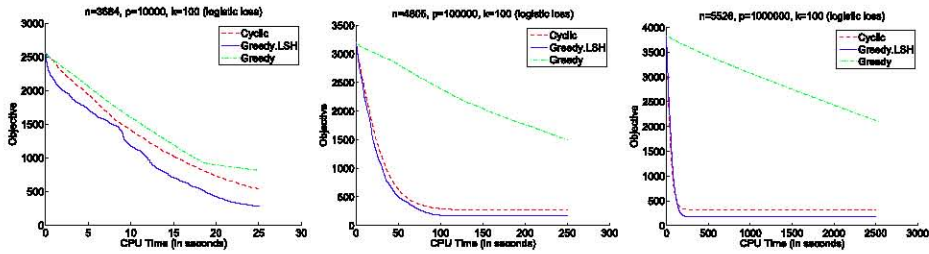

Figure 1: (*best viewed in color*) Objective vs. CPU time plots for logistic loss using $p = 10^4, 10^5, 10^6$

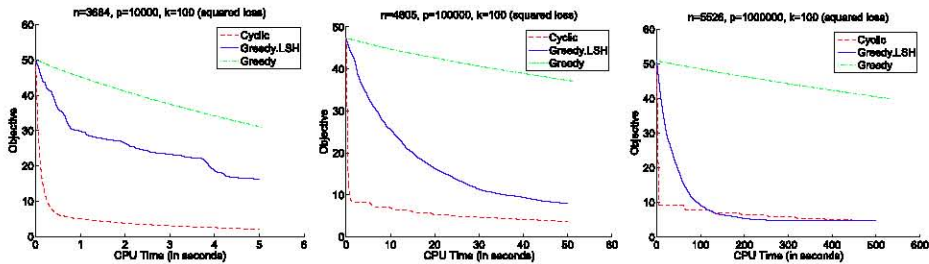

Figure 2: (*best viewed in color*) Objective vs. CPU time plots for squared loss using $p = 10^4, 10^5, 10^6$

demonstrated by these plots. In contrast with the logistic case, here cyclic as able to finish several full sweeps through the $p$ coordinate, namely $13.4, 10.5$ and $7.9$ sweeps for $p = 10^4, 10^5$ and $10^6$ respectively. even though cyclic got lower objective values, it was at the expense of sparsity: cylic's final iterates were usually 10 times denser than those of greedy.LSH.

Figure 3 shows the plots for the objective versus number of coordinate descent steps. We clearly see that cyclic is wasteful in terms of number of coordinate updates and greedy achieves much greater descent in the objective per coordinate update. Moreover, greedy.LSH is much closer to greedy in its per coordinate-update performance (to the extent that it is hard to tell them apart in some of these plots). This plot thus suggests the improvements possible with better nearest-neighbor implementations that perform the greedy step even faster than our non-optimized greedy.LSH implementation.

Cyclic coordinate descent is one of the most competitive methods for large scale $\ell_1$-regularized problems [9]. We are able to outperform it for large problems using a homegrown implementation that was not optimized for performance. This provides strong reasons to believe that with a careful well-tuned LSH implementation, and indeed with better data structures than LSH, nearest neighbor based greedy methods should be able to scale to problems beyond the reach of current methods.

### Acknowledgments

We gratefully acknowledge the support of NSF under grants IIS-1018426 & CCF-0728879. ISD acknowledges support from the Moncrief Grand Challenge Award.

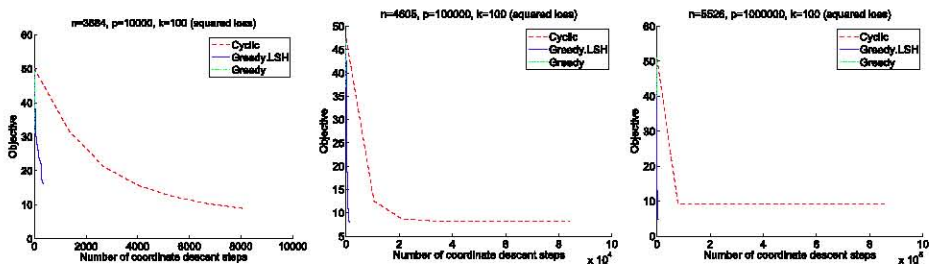

Figure 3: (*best viewed in color*) Objective vs. no. of coordinate updates: squared loss using $p = 10^4, 10^5, 10^6$

# References

[1] N. Ailon and B. Chazelle. Approximate nearest neighbors and the fast johnson-lindenstrauss transform. In *Proc. 38th STOC*, pages 557–563, 2006.

[2] S. Arya and D. M. Mount. Approximate nearest neighbor queries in fixed dimensions. In *Proc. 4th ACM-SIAM SODA*, pages 271–280, 1993.

[3] S. Arya, T. Malamatos, and D. M. Mount. Space-time tradeoffs for approximate nearest neighbor searching. *Journal of the ACM*, 57(1), 2009.

[4] D.P. Bertsekas. *Nonlinear programming*. Athena Scientific, Belmont, MA, 1995.

[5] E. Candes and T. Tao. The Dantzig selector: Statistical estimation when $p$ is much larger than $n$. *Annals of Statistics*, 2006.

[6] Y. Censor and S. A. Zenios. *Parallel optimization: Theory, algorithms, and applications*. Oxford University Press, 1997.

[7] I. Daubechies, M. Defrise, and C. De Mol. An iterative thresholding algorithm for linear inverse problems with a sparsity constraint. *Comm. Pure Appl. Math.*, 57(11):1413–1457, 2004.

[8] J. Friedman, T. Hastie, H. Hofling, and R. Tibshirani. Pathwise coordinate optimization. 2007.

[9] Jerome Friedman, Trevor Hastie, and Robert Tibshirani. Regularization paths for generalized linear models via coordinate descent. *Journal of Statistical Software*, 33(1):1–22, 2010.

[10] A. Genkin, D. D. Lewis, and D. Madigan. Large-scale bayesian logistic regression for text categorization. *Technometrics*, 49(3):291–304, 2007.

[11] S. Har-Peled. Lectures notes on geometric approximation algorithms. 2009. URL `http://valis.cs.uiuc.edu/~sariel/teach/notes/aprx/lec/`.

[12] P. Indyk and R. Motwani. Approximate nearest neighbors: towards removing the curse of dimensionality. In *Proc. 30th STOC*, pages 604–613, 1998.

[13] A. Saha and A. Tewari. On the finite time convergence of cyclic coordinate descent methods. preprint, 2010.

[14] S. Shalev-Shwartz and A. Tewari. Stochastic methods for $\ell_1$ regularized loss minimization. In *ICML*, 2009.

[15] J. Tropp. Just relax: Convex programming methods for identifying sparse signals in noise. *IEEE Trans. Info Theory*, 52(3):1030–1051, March 2006.

[16] P. Tseng and S. Yun. A block-coordinate gradient descent method for linearly constrained nonsmooth separable optimization. *Journal of Optimization Theory and Applications*, 140(3): 513–535, .

[17] P. Tseng and S. Yun. A coordinate gradient descent method for nonsmooth separable minimization. *Math. Prog. B*, 117:387–423, .

[18] M. J. Wainwright. Sharp thresholds for noisy and high-dimensional recovery of sparsity using $\ell_1$-constrained quadratic programming (lasso). *IEEE Transactions on Info. Theory*, 55:2183–2202, 2009.

[19] T. T. Wu and K. Lange. Coordinate descent algorithms for lasso penalized regression. *Annals of Applied Statistics*, 2:224–244, 2008.

